# Simultaneous Object Detection and Ranking with Weak Supervision

Matthew B. Blaschko        Andrea Vedaldi        Andrew Zisserman

Department of Engineering Science
University of Oxford
United Kingdom

## Abstract

A standard approach to learning object category detectors is to provide strong supervision in the form of a region of interest (ROI) specifying each instance of the object in the training images [17]. In this work are goal is to learn from heterogeneous labels, in which some images are only weakly supervised, specifying only the presence or absence of the object or a weak indication of object location, whilst others are fully annotated.

To this end we develop a discriminative learning approach and make two contributions: (i) we propose a structured output formulation for weakly annotated images where full annotations are treated as latent variables; and (ii) we propose to optimize a ranking objective function, allowing our method to more effectively use negatively labeled images to improve detection average precision performance.

The method is demonstrated on the benchmark INRIA pedestrian detection dataset of Dalal and Triggs [14] and the PASCAL VOC dataset [17], and it is shown that for a significant proportion of weakly supervised images the performance achieved is very similar to the fully supervised (state of the art) results.

## 1   Introduction

Learning from weakly annotated data is a long standing goal for the practical application of machine learning techniques to real world data. Expensive manual labeling steps should be avoided if possible, while weakly labeled and unlabeled data sources should be exploited in order to improve performance with little to no additional cost. In this work, we propose a unified framework for learning to detect objects in images from data with heterogeneous labels. In particular, we consider the case of image collections for which we would like to predict bounding box localizations, but that (for a significant proportion of the training data) only image level binary annotations are provided indicating the presence or absence of an object, or that weak indications of object location are given without a precise bounding box annotation.

We approach this task from the perspective of structured output learning [3, 35, 36], building on the approach of Blaschko and Lampert [8], in which a structured output support vector machine formulation [36] is used to directly learn a regressor from images to object localizations parameterized by the coordinates of a bounding box. We extend this framework here to weakly annotated images by treating missing information in a latent variable fashion following [2, 40]. Available annotation, such as the presence or absence of an object in an image, constrains the set of values the latent variable can take. In the case that complete label information is provided [40] reduces to [36], giving a unified framework for data with heterogeneous levels of annotation. We empirically observe that the localization approach of [8] fails in the case that there are many images with no object present, motivating a slight modification of the learning algorithm to optimize detection *ranking* analogous

to [11, 21, 41]. We extend these works to the case that the predictions to be ranked are structured outputs. When combined with discriminative latent variable learning, this results in an algorithm similar to multiple instance ranking [6], but we exploit the full generality of structured output learning.

The computer vision literature has approached learning from weakly annotated data in many different ways. Search engine results [20] or associated text captions [5, 7, 13, 34] are attractive due to the availability of millions of tagged or captioned images on the internet, providing a weak form of labels beyond unsupervised learning [37]. This generally leads to ambiguity as captions tend to be correlated with image content, but may contain errors. Alternatively, one may approach the problem of object detection by considering generic properties of objects or their attributes in order to combine training data from multiple classes [1, 26, 18]. Deselaers et al. learn the common appearance of multiple object categories, which yields an estimate of where in an image an object is without specifying the specific class to which it belongs [15]. This can then be utilized in a weak supervision setting to learn a detector for a specific object category. Carbonetto et al. consider a Bayesian framework for learning across incomplete, noisy, segmentation-level annotation [10]. Structured output learning with latent variables has been proposed for inferring partial truncation of detections due to occlusion or image boundaries [38]. Image level binary labels have often been used, as this generally takes less time for a human annotator to produce [4, 12, 23, 28, 30, 31, 33]. Here, we consider this latter kind of weak annotation, and will also consider cases where the object center is constrained to a region in the image, but that exact coordinates are not given [27]. Simultaneous localization and classification using a discriminative latent variable model has been recently explored in [29], but that work has not considered mixed annotation, or a structured output loss.

The rest of this paper is structured as follows. In Section 2 we review a structured output learning formulation for object detection that will form the basis of our optimization. We then propose to improve that approach to better handle negative training instances by developing a ranking objective in Section 3. The resulting objective allows us to approach the problem of weakly annotated data in Section 4, and the methods are empirically validated in Section 5.

## 2 Object Detection with Structured Output Learning

Structured output learning generalizes traditional learning settings to the prediction of more complex output spaces, in which there may be non-trivial interdependencies between components of the output. In our case, we would like to learn a mapping $f : \mathcal{X} \to \mathcal{Y}$ where $\mathcal{X}$ the space of images and $\mathcal{Y}$ is the space of bounding boxes or no bounding box: $\mathcal{Y} \equiv \emptyset \bigcup (l, t, r, b)$, where $(l, t, r, b) \in \mathbb{R}^4$ specifies the left, top, right, and bottom coordinates of a bounding box. This approach was first proposed by [8] using the Structured Output SVM formulation of [36]:

$$\min_{w, \xi} \quad \frac{1}{2}\|w\|^2 + C\frac{1}{n}\sum_i \xi_i \tag{1}$$

$$\text{s.t.} \quad \langle w, \phi(x_i, y_i)\rangle - \langle w, \phi(x_i, y)\rangle \geq \Delta(y_i, y) - \xi_i, \quad \forall i, y \in \mathcal{Y} \setminus \{y_i\} \tag{2}$$

$$\xi_i \geq 0 \quad \forall i \tag{3}$$

where $\Delta(y_i, y)$ is a loss for predicting $y$ when the true output is $y_i$, and $\phi(x_i, y_i)$ is a *joint kernel map* that measures statistics of the image, $x_i$, local to the bounding box, $y_i$ [8, 9].[1] Training is achieved using delayed constraint generation, and at test time, a prediction is made by computing $f(x) = \text{argmax}_y \langle w, \phi(x, y)\rangle$.

It was proposed in [8] to treat images in which there is no instance of the object of interest as zero vectors in the Hilbert space induced by $\phi$, i.e. $\phi(x, y_-) = 0 \; \forall x$ where $y_-$ indicates the label that there is no object in the image (i.e. $y_- \equiv \emptyset$). During training, constraints are generated by finding $\tilde{y}_i^* = \text{argmax}_{y \in \mathcal{Y} \setminus \{y_i\}} \langle w, \phi(x_i, y)\rangle + \Delta(y_i, y)$. For negative images, $\Delta(y_-, y) = 1$ if $y$ indicates an object is present, so the maximization corresponds simply to finding the bounding box with highest score. The resulting constraint corresponds to:

$$\xi_i \geq 1 + \langle w, \phi(x_i, \tilde{y}_i^*)\rangle \tag{4}$$

which tends to decrease the score associated with all bounding boxes in the image. The primary problem with this approach is that it optimizes a regularized risk functional for which negative images are treated equally with positive images. In the case of imbalances in the training data where a large majority of images do not contain the object of interest, the objective function may be dominated by the terms in $\sum_i \xi_i$ for which there is no bounding box present. The learning procedure may focus on decreasing the score of candidate detections in negative images rather than on increasing the score of correct detections. We show empirically in Section 5 that this treatment of negative images is in fact detrimental to localization performance. The results presented in [8] were achieved by training only on images with an instance of the object present, ignoring large quantities of negative training data. Although one may attempt to address this problem by adjusting the loss function, $\Delta$, to penalize negative images less than positive images, this approach is heuristic and requires searching over an additional parameter during training (the relative size of the loss for negative images). We address this imbalance more elegantly without introducing additional parameters in the following section.

## 3   Learning to Rank

We propose to remedy the shortcomings outlined in the previous section by modifying the objective in Equation (1) to simultaneously localize and rank object detections. The following constraints applied to the test set ensure a perfect ranking, that is that every true detection has a higher score than all false detections:

$$\langle w, \phi(x_i, y_i) \rangle > \langle w, \phi(x_j, \tilde{y}_j) \rangle \quad \forall i, j, \tilde{y}_j \in \mathcal{Y} \setminus \{y_j\}. \tag{5}$$

We modify these constraints, incorporating a structured output loss, in the following structured output ranking objective

$$\min_{w, \xi} \quad \frac{1}{2} \|w\|^2 + C \frac{1}{n \cdot n_+} \sum_{i,j} \xi_{ij} \tag{6}$$

$$\text{s.t.} \quad \langle w, \phi(x_i, y_i) \rangle - \langle w, \phi(x_j, \tilde{y}_j) \rangle \geq \Delta(y_j, \tilde{y}_j) - \xi_{ij} \quad \forall i, j, \tilde{y}_j \in \mathcal{Y} \setminus \{y_j\} \tag{7}$$

$$\xi_{ij} \geq 0 \quad \forall i, j \tag{8}$$

where $n_+$ denotes the number of positive instances in the training set. As compared with Equations (1)-(3), we now compare each positive instance to *all bounding boxes in all images* in the training set instead of just the bounding boxes from the image it comes from. The constraints attempt to give all positive instances a score higher than all negative instances, where the size of the margin is scaled to be proportional to the loss achieved by the negative instance. We note that one can use this same approach to optimize related ranking objectives, such as precision at a given detection rate, by extending the formulations of [11, 41] to incorporate our structured output loss function, $\Delta$.

As in [8, 36] we have an intractable number of constraints in Equation (7). We will address this problem using a constraint generation approach with a 1-slack formulation

$$\min_{w, \xi} \quad \frac{1}{2} \|w\|^2 + C\xi \tag{9}$$

$$\text{s.t.} \quad \sum_{ij} \langle w, \phi(x_i, y_i) \rangle - \langle w, \phi(x_j, \tilde{y}_j) \rangle \geq \sum_{ij} \Delta(y_j, \tilde{y}_j) - \xi \quad \forall \tilde{\mathbf{y}} \in \bigoplus_j \mathcal{Y} \setminus \{y_j\} \tag{10}$$

$$\xi \geq 0 \tag{11}$$

where $\tilde{\mathbf{y}}$ is a vector with $j$th element $\tilde{y}_j$. Although this results in a number of constraints exponential in the number of training examples, we can solve this efficiently using a cutting plane algorithm. The proof of equivalence between this optimization problem and that in Equations (6)-(8) is analogous to the proof in [22, Theorem 1]. We are only left to find the maximally violated constraints in Equation (10). Algorithm 1 gives an efficient procedure for doing so.

Algorithm 1 works by first scoring all positive regions, as well as finding and scoring the maximally violated regions from each image. We make use of the transitivity of ordering these two sets of scores to avoid comparing all pairs in a naïve fashion. If $\langle w, \phi(x_j, \tilde{y}_j^*) \rangle \geq \langle w, \phi(x_i, y_i) \rangle$ and

**Algorithm 1** 1-slack structured output ranking – maximally violated constraint.

---
**Ensure:** Maximally violated constraint is $\delta - \langle w, \psi \rangle \leq \xi$
    **for all** $i$ **do**
        $s_i^+ = \langle w, \phi(x_i, y_i) \rangle$
    **end for**
    **for all** $j$ **do**
        $\tilde{y}_j^* = \mathrm{argmax}_y \langle w, \phi(x_j, y) \rangle + \Delta(y_j, y)$
        $s_j^- = \langle w, \phi(x_j, \tilde{y}_j^*) \rangle + \Delta(y_j, \tilde{y}_j^*)$
    **end for**
    $(s^+, p^+) = \mathrm{sort}(s^+)$ {$p^+$ is a vector of indices specifying a given score's original index.}
    $(s^-, p^-) = \mathrm{sort}(s^-)$
    $\delta = 0, k = 1, \psi = \phi_+ = \mathbf{0}$
    **for all** $j$ **do**
        **while** $s_j^- > s_k^+ \wedge k \leq n_+ + 1$ **do**
            $\phi_+ = \phi_+ + \phi\left(x_{p_k^+}, y_{p_k^+}\right)$
            $k = k + 1$
        **end while**
        $\psi = \psi + \phi_+ - (k-1)\phi\left(x_{p_j^-}, \tilde{y}_{p_j^-}^*\right)$
        $\delta = \delta + (k-1)\Delta(y_j, \tilde{y}_j^*)$
    **end for**

---

$\langle w, \phi(x_i, y_i) \rangle \geq \langle w, \phi(x_p, y_p) \rangle$, we do not have to compare $\langle w, \phi(x_j, \tilde{y}_j^*) \rangle$ and $\langle w, \phi(x_p, y_p) \rangle$. Instead, we sort the instances of the class by their score, and sort the negative instances by their score as well. We keep an accumulator vector for positive images, $\phi_+$, and a count of the number of violated constraints ($k - 1$). We iterate through each violated region, ordered by score, and sum the violated constraints into $\psi$ and $\delta$, yielding the maximally violated 1-slack constraint.

## 4 Weakly Supervised Data

Now that we have developed a structured output learning framework that is capable of appropriately handling images from the background class, we turn our attention to the problem of learning with weakly annotated data. We will consider the problem in full generality by assuming that we have bounding box level annotation for some training images, but only binary labels or weak location information for others. For negatively labeled images, we know that no bounding box in the entire image contains an instance of the object class, while for positive images at least one bounding box belongs to the class of interest. We approach this issue by considering the location of a bounding box to be a latent variable to be inferred during training. The value that this variable can take is constrained by the weak annotation. In the case that we have only a binary image-level label, we constrain the latent variable to indicate that some region of the image corresponds to the object of interest. In a more constrained case, such as annotation indicating the object center, we constrain the latent variable to belong to the set of bounding boxes that have a center consistent with the annotation. There is an asymmetry in the image level labeling in that negative labels can be considered to be full annotation (i.e. all bounding boxes do not contain an instance of the object), while positive labels are incomplete.[2] We consider the index variable $j$ to range over all completely labeled images, including negative images.

We consider a modification of the constrained objective developed in the previous section to include constraints of the form given in Equation (7), but also constraints for our weakly annotated positive images, which we index by $m$,

$$\left(\max_{\hat{y}_m \in \mathcal{Y}_m} \langle w, \phi(x_m, \hat{y}_m) \rangle\right) - \langle w, \phi(x_j, \tilde{y}_j) \rangle \geq \Delta(y_j, \tilde{y}_j) - \xi_{mj} \quad \forall m, j, \tilde{y}_j \in \mathcal{Y} \setminus \{y_j\}, \quad (12)$$

where $\mathcal{Y}_m$ is the set of bounding boxes consistent with the weak annotation for image $m$. Due to the maximization over $\hat{y}_m$, the optimization is no longer convex, but we can find a local optimum using the CCCP algorithm [40]. This is effectively equivalent to the case of loss-rescaled multiple instance learning, and we note that the resulting objective has similarities to that of [2]. Viewed another way, we treat the location of the hypothesized bounding box as a latent variable. In order to use this in our discriminative optimization, we will try to put a large margin between the maximally scoring box and all bounding boxes with high loss. Though our algorithm does not have direct information about the true location of the object of interest, it tries to learn a discriminant function that can distinguish a region in the positively labeled images from all regions in the negatively labeled images.

## 5   Results

We validate our model on the benchmark INRIA pedestrian detection dataset of Dalal and Triggs [14] using a histogram of oriented gradients (HOG) representation, and the PASCAL VOC dataset [16, 17]. Following [9, 24, 25], we provide detailed results on the `cat` class as the high variation in pose is appropriate for testing a bag of words model, but also provide summary results for all classes in the form of improvement in mean average precision (mean AP). We first illustrate the performance of the ranking objective developed in Section 3 and subsequently show the performance of learning with weakly supervised data using the latent variable approach of Section 4.

### 5.1   Experimental Setup

We have implemented variants of two popular object detection systems in order to show the generalization of the approaches developed in this work to different levels of supervision and feature descriptors. In the first variant, we have used a linear bag of words model similar to that developed in [8, 24, 25]. Inference of maximally violated constraints and object detection was performed using Efficient Subwindow Search (ESS) branch-and-bound inference [24, 25]. The joint kernel map, $\phi$, was constructed using a concatenation of the bounding box visual words histogram (the *restriction kernel*) and a global image histogram, similar to the approach described in [9]. Results are presented on the VOC 2007 dataset [16, 17].

The second variant of the detector is based on the histogram of oriented gradients (HOG) representation [14]. HOG subdivides the image into cells, usually of size $8 \times 8$ pixels, and computes for each cell a weighed histogram of the gradient orientations. The experiments use the HOG variant of [19], which results in a 31-dimensional histogram for each cell. The HOG features are extracted at multiple scales, forming a pyramid. An object is described by a rectangular arrangement of HOG cells (the aspect ratio of the rectangular grouping is fixed). The joint feature map, $\phi$, extracts from the HOG representation of the image the rectangular group of HOG cells at a given scale and location [38]. A constant bias term is appended to the resulting feature [38] for all but the ranking cost functional, as the bias term cancels out in that formulation. Note that the model is analogous to the HOG detector of [14], and in particular does *not* use flexible parts as in [19]. Results are presented for the INRIA pedestrian data set [14].

### 5.2   Learning to Rank

In order to evaluate the effects of optimizing the ranking objective developed in this work, we begin by comparing the performance of the objective in Equations (6)-(8) in a fully supervised setting with that of the objective in Equations (1)-(3), which correspond to the optimization proposed in [8].

In Figure 1, we show the relative performance of the linear bag of visual words model applied to the PASCAL VOC 2007 data set [17]. We first show results for the `cat` class in which $10\%$ of negative images are included in the training set (Figure 1(a)), and subsequently results for which all negative images are used for training (Figure 1(b)). While the ranking objective can appropriately handle varying amounts of negative training data, the objective in Equation (1) fails, resulting in worse performance as the amount of negative training data increases. These results empirically show the shortcomings of the treatment of negative images proposed in [8], but the ranking objective by contrast is robust to large imbalances between positive and negative images. Mean AP increases by $69\%$ as a result of using the ranking objective when $10\%$ of negative images are included during training, and mean AP improves by $71\%$ when all negative images are used.

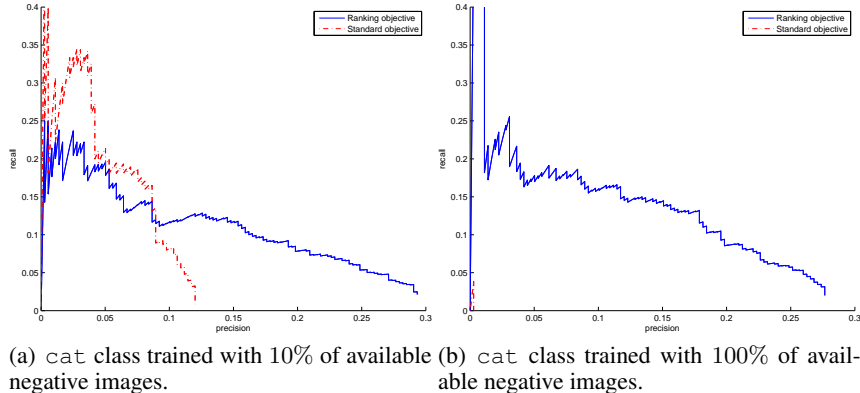

(a) `cat` class trained with 10% of available negative images.

(b) `cat` class trained with 100% of available negative images.

Figure 1: Precision-recall curves for the structured output ranking objective proposed in this paper (blue) vs. the structured output objective proposed in [8] (red) for varying amounts of negative training data. Results are shown on the `cat` class from the PASCAL VOC 2007 data set for 10% of negative images (1(a)) and for 100% of negatives (1(b)). In all cases a linear bag of visual words model was employed (see text for details). The structured output objective proposed in [8] performs worse with increasing amounts of negative training data, and the algorithm completely fails in 1(b). The ranking objective, on the other hand, does not suffer from this shortcoming (blue curves).

Figure 2.(a) analyzes the performance of the HOG pedestrian detection on the INRIA data set. Three cost functionals are compared: a simple binary SVM, the structural SVM model of (1), and the ranking SVM model of (6). The INRIA dataset contains 1218 negative images (i.e. images not containing people). Each image is subdivided (in scale and space) into twenty sub-images and a maximally violating window (object location) is extracted from each of those. This results in 24360 negative windows. The dataset contains also 612 positive images, for a total of 1237 labeled pedestrians. Thus there are about twenty times more negative examples than positive ones. Reweighted versions of the binary and structural SVM models that balance the number of positive and negative examples are also tested. As the figure shows, balancing the data in the cost functional is important, especially for the binary SVM model; the ranking model is slightly superior to the other formulations, with average precision of 77%, and does not require an adjustment to the loss to account for a given level of data imbalance. By comparison, the state-of-the-art detector of [32] has average precision 78%. We conjecture that this small difference in performance is due to their use of color information.

## 5.3 Learning with Weak Annotations

To evaluate the objective in the case of weak supervision, we have additionally performed experiments in which we have varied the percentage of bounding box annotations provided to the learning algorithm.

Figure 3 contrasts the performance on the VOC dataset of our proposed discriminative latent variable algorithm with that of a fully supervised algorithm in which weakly annotated training data are ignored. We have run the algorithm for 10% of images having full bounding box annotations (with the other 90% weakly labeled) and for 50% of images having complete annotation. In the fully supervised case, we ignore all images that do not have full bounding box annotation and train the fully supervised ranking objective developed in Section 3. In all cases, the latent variable model performs convincingly better than subsampling. For 10% of images fully annotated, mean AP increases by 64%, and with 50% of images fully annotated, mean AP increases by 83%.

Figure 2.(b) reports the performance of the latent variable ranking model (8) for the HOG-based detector on the INRIA pedestrian dataset. Only *one* positive image is fully labeled with the pedestrian bounding boxes while the remaining positive images are weakly labeled. Since most positive images contain multiple pedestrians, the weak annotations carry a minimal amount of information that is still sufficient to distinguish the different pedestrian instances. Specifically, the bounding boxes are discarded and only their centers are kept. Estimating the latent variables consists of a search over

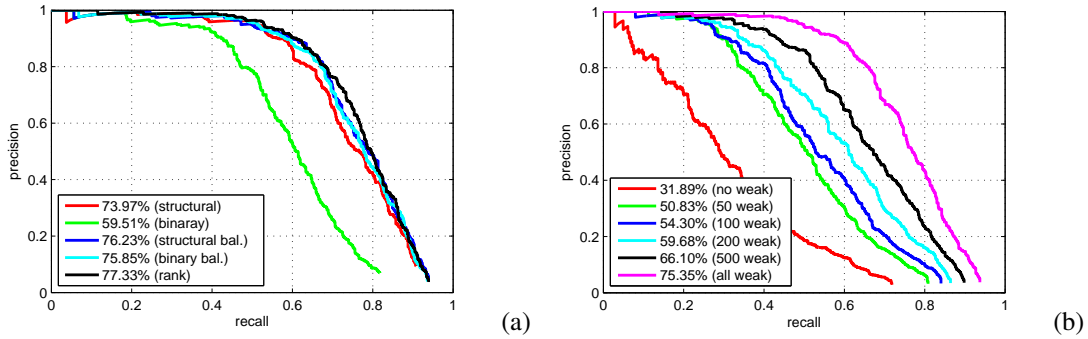

(a)

(b)

Figure 2: **(a)** Precision-recall curves for different formulations: binary and structural SVMs, balanced binary and structural SVMs, ranking SVM. The unbalanced SVMs, and in particular the binary one, do not work well due to the large number of negative examples compared to the positive ones. The ranking formulation is slightly better than the other balanced costs for this dataset. **(b)** Precision-recall curves for increasing amounts of weakly supervised data for the ranking formulation. For all curves, only one image is fully labeled with bounding boxes around pedestrians, while the other images are labeled only by the pedestrian centers. The first curve (AP 32%) corresponds to the case in which only the fully supervised image is used; the last curve (AP 75%) to the case in which all the other training images are added with weak annotations. The performance is almost as good as the fully supervised case (AP 77%) of (a).

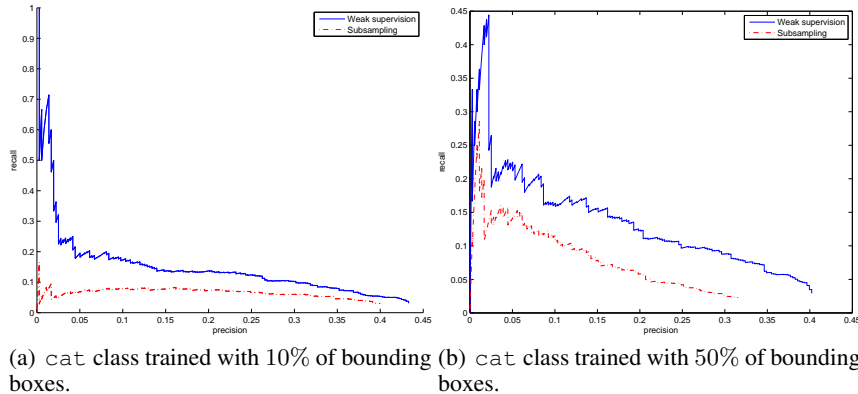

(a) `cat` class trained with 10% of bounding boxes.  (b) `cat` class trained with 50% of bounding boxes.

Figure 3: Precision-recall curves for the structured output ranking objective proposed in this paper trained with a linear bag of words image representation and weak supervision (blue) vs. only using fully labeled samples (red). Results are shown for 10% of bounding boxes (left) and for 50% of bounding boxes (right), the remainder of the images were provided with weak annotation indicating the presence or absence of an object in the image, but not the object location. In both cases, the latent variable model (blue) results in performance that is substantially better than discarding weakly annotated images and using a fully supervised setting (red).

all object locations and scales for which the corresponding bounding box center is within a given bound of the labeled center (the bound is set to 25% of the length of the box diagonal). In other words, a weak annotation contains only approximate location information. This gives robustness to inaccuracies in manually labeling the centers. The figure shows how the model performs when, in addition to the singly fully annotated image, an increasing number of weakly annotated images are added. Starting from 32% AP, the method improves up to 75% AP, which is remarkably similar to the best result (77% AP) obtained with full supervision.

# 6 Discussion

We can draw several conclusions from the results in Section 5. First, using the learning formulation developed in [8], negative images are not handled properly, resulting in the undesired behavior that additional negative images in the training data decrease performance. The special case of the objective in Equations (1)-(3), for which no negative training data are incorporated, can be viewed roughly as an estimate of the $\log$ probability of an object being present at a location conditioned on that an object is present in the image. While this results in reasonable performance in terms of recall (c.f. [8]), it does not result in a good average precision (AP) score. In fact, the results presented in [8] were computed by training the objective function only on positive images, and then using a separate non-linear ranking function based on global image statistics. Using only positively labeled images in the objective presented in Section 2 only incorporates a subset of the constraints in Equation (7) corresponding to $i = j$. Incorporating all these constraints directly optimizes ranking, enabling the use of all available negative training data to improve localization performance.

Reweighting the loss corresponding to positive and negative examples resulted in similar performance to the ranking objective on the INRIA pedestrian data set, but requires a search across an additional parameter. From the perspective of regularized risk, subsampling negative images can be viewed as a noisy version of this reweighting, and experiments on PASCAL VOC using the objective in (1) showed poor performance over a wide range of sampling rates. The ranking objective by contrast weights loss from the negative examples appropriately (Algorithm 1) according to their contribution to the loss for the precision-recall curve. This is a much more principled and robust criterion for setting the loss function.

By using the ranking objective to treat negative images, learning with weak annotations was made directly applicable using a discriminative latent variable model. Results showed consistent improvement across different proportions of weakly and fully supervised data. Our formulation handled different ratios of weakly annotated and fully annotated training data without additional parameter tuning in the loss function. The discriminative latent variable approach has been able to achieve performance within a few percent of that achieved by a fully supervised system using *only one* fully supervised label. The weak labels used for the remaining data are significantly less expensive to supply [39]. That this is consistent across the data sets reported here indicates that discriminative latent variable models are a promising strategy for treating weak annotation in general.

**Acknowledgments**

The first author is supported by the Royal Academy of Engineering through a Newton International Fellowship. The research leading to these results has received funding from the European Research Council under the European Community's Seventh Framework Programme (FP7/2007- 2013) / ERC grant agreement no. 228180, and from the PASCAL2 network of excellence.

## Footnotes

[1]As in [8], we make use of the margin rescaling formulation of structured output learning. The slack rescaling variant is equally applicable [36].

[2]Note that this is exactly the asymmetry discussed in [2] in the context of multiple instance learning. Our setting can be seen as a generalization to mixed annotations.

# References

[1] B. Alexe, T. Deselaers, and V. Ferrari. What is an object? In *Proceedings of the IEEE Conference on Computer Vision and Pattern Recognition*, June 2010.

[2] S. Andrews, I. Tsochantaridis, and T. Hofmann. Support vector machines for multiple-instance learning. In *Advances in Neural Information Processing Systems*, pages 561–568. MIT Press, 2003.

[3] G. H. Bakır, T. Hofmann, B. Schölkopf, A. J. Smola, B. Taskar, and S. V. N. Vishwanathan. *Predicting Structured Data*. MIT Press, 2007.

[4] A. Bar Hillel, T. Hertz, and D. Weinshall. Efficient learning of relational object class models. In *Proceedings of the International Conference on Computer Vision*, pages 1762–1769, 2005.

[5] T. Berg, A. Berg, J. Edwards, M. Mair, R. White, Y. Teh, E. Learned-Miller, and D. Forsyth. Names and Faces in the News. In *Proceedings of the IEEE Conference on Computer Vision and Pattern Recognition, Washington, DC*, 2004.

[6] C. Bergeron, J. Zaretzki, C. Breneman, and K. P. Bennett. Multiple instance ranking. In *Proceedings of the International Conference on Machine Learning*, pages 48–55, 2008.

[7] M. B. Blaschko and C. H. Lampert. Correlational spectral clustering. In *Proceedings of the IEEE Conference on Computer Vision and Pattern Recognition*, 2008.

[8] M. B. Blaschko and C. H. Lampert. Learning to localize objects with structured output regression. In *Proceedings of the European Conference on Computer Vision*, 2008.

[9] M. B. Blaschko and C. H. Lampert. Object localization with global and local context kernels. In *Proceedings of the British Machine Vision Conference*, 2009.

[10] P. Carbonetto, G. Dorkó, C. Schmid, H. Kück, and N. Freitas. Learning to recognize objects with little supervision. *International Journal of Computer Vision*, 77(1–3):219–237, 2008.

[11] O. Chapelle and S. S. Keerthi. Efficient algorithms for ranking with svms. *Information Retrieval*, 2009.

[12] O. Chum and A. Zisserman. An exemplar model for learning object classes. In *Proceedings of the IEEE Conference on Computer Vision and Pattern Recognition*, 2007.

[13] T. Cour, B. Sapp, C. Jordan, and B. Taskar. Learning from ambiguously labeled images. In *Proceedings of the IEEE Conference on Computer Vision and Pattern Recognition*, 2009.

[14] N. Dalal and B. Triggs. Histogram of Oriented Gradients for Human Detection. In *Proceedings of the IEEE Conference on Computer Vision and Pattern Recognition*, volume 2, pages 886–893, 2005.

[15] T. Deselaers, B. Alexe, and V. Ferrari. Localizing objects while learning their appearance. In *Proceedings of the European Conference on Computer Vision*, 2010.

[16] M. Everingham, L. Van Gool, C. K. I. Williams, J. Winn, and A. Zisserman. The PASCAL Visual Object Classes Challenge 2007 (VOC2007) Results. http://www.pascal-network.org/challenges/VOC/voc2007/workshop/index.html, 2007.

[17] M. Everingham, L. Van Gool, C. K. I. Williams, J. Winn, and A. Zisserman. The pascal visual object classes (voc) challenge. *International Journal of Computer Vision*, 88(2):303–338, June 2010.

[18] A. Farhadi, I. Endres, D. Hoiem, and D. Forsyth. Describing objects by their attributes. *Proceedings of the IEEE Conference on Computer Vision and Pattern Recognition*, pages 1778–1785, 2009.

[19] P. Felzenszwalb, D. Mcallester, and D. Ramanan. A discriminatively trained, multiscale, deformable part model. In *Proceedings of the IEEE Conference on Computer Vision and Pattern Recognition*, 2008.

[20] R. Fergus, L. Fei-Fei, P. Perona, and A. Zisserman. Learning object categories from Google's image search. In *Proceedings of the International Conference on Computer Vision*, 2005.

[21] T. Joachims. Optimizing search engines using clickthrough data. In *KDD '02: Proceedings of the eighth ACM SIGKDD international conference on Knowledge discovery and data mining*, pages 133–142, New York, NY, USA, 2002. ACM.

[22] T. Joachims, T. Finley, and C.-N. J. Yu. Cutting-plane training of structural svms. *Machine Learning*, 77(1):27–59, 2009.

[23] G. Kim and A. Torralba. Unsupervised detection of regions of interest using iterative link analysis. In Y. Bengio, D. Schuurmans, J. Lafferty, C. K. I. Williams, and A. Culotta, editors, *Advances in Neural Information Processing Systems*, pages 961–969. 2009.

[24] C. H. Lampert, M. B. Blaschko, and T. Hofmann. Beyond sliding windows: Object localization by efficient subwindow search. *Proceedings of the IEEE Conference on Computer Vision and Pattern Recognition*, 2008.

[25] C. H. Lampert, M. B. Blaschko, and T. Hofmann. Efficient subwindow search: A branch and bound framework for object localization. *IEEE Transactions on Pattern Analysis and Machine Intelligence*, 2009.

[26] C. H. Lampert, H. Nickisch, and S. Harmeling. Learning to detect unseen object classes by between-class attribute transfer. In *Proceedings of the IEEE Conference on Computer Vision and Pattern Recognition*, pages 951–958, 2009.

[27] B. Leibe, A. Leonardis, and B. Schiele. Combined object categorization and segmentation with an implicit shape model. In *Workshop on Statistical Learning in Computer Vision, ECCV*, May 2004.

[28] F. Moosmann, D. Larlus, and F. Jurie. Learning saliency maps for object categorization. In *ECCV International Workshop on The Representation and Use of Prior Knowledge in Vision*, 2006.

[29] M. H. Nguyen, L. Torresani, F. De la Torre Frade, and C. Rother. Weakly supervised discriminative localization and classification: A joint learning process. In *Proceedings of the International Conference on Computer Vision*, 2009.

[30] A. Opelt, A. Fussenegger, A. Pinz, and P. Auer. Weak hypotheses and boosting for generic object detection and recognition. In *Proceedings of the 8th European Conference on Computer Vision, Prague, Czech Republic*, volume 2, pages 71–84, 2004.

[31] A. Opelt and A. Pinz. Object localization with boosting and weak supervision for generic object recognition. In *Scandinavian Conference on Image Analysis*, pages 862–871, 2005.

[32] P. Ott and M. Everingham. Implicit color segmentation features for pedestrian and object detection. In *Proceedings of the International Conference on Computer Vision*, 2009.

[33] C. Pantofaru and M. Hebert. A framework for learning to recognize and segment object classes using weakly supervised training data. In *Proceedings of the British Machine Vision Conference*, 2007.

[34] N. Rasiwasia and N. Vasconcelos. Scene classification with low-dimensional semantic spaces and weak supervision. In *Proceedings of the IEEE Conference on Computer Vision and Pattern Recognition*, 2008.

[35] B. Taskar, C. Guestrin, and D. Koller. Max-margin markov networks. In S. Thrun, L. Saul, and B. Schölkopf, editors, *Advances in Neural Information Processing Systems*. 2004.

[36] I. Tsochantaridis, T. Hofmann, T. Joachims, and Y. Altun. Support vector machine learning for interdependent and structured output spaces. In *Proceedings of the International Conference on Machine Learning*, 2004.

[37] T. Tuytelaars, C. H. Lampert, M. B. Blaschko, and W. Buntine. Unsupervised object discovery: A comparison. *International Journal of Computer Vision*, 88(2):61–85, 2010.

[38] A. Vedaldi and A. Zisserman. Structured output regression for detection with partial truncation. In *Advances in Neural Information Processing Systems*, 2009.

[39] S. Vijayanarasimhan and K. Grauman. Multi-level active prediction of useful image annotations for recognition. In D. Koller, D. Schuurmans, Y. Bengio, and L. Bottou, editors, *Advances in Neural Information Processing Systems*, pages 1705–1712. 2009.

[40] C.-N. J. Yu and T. Joachims. Learning structural svms with latent variables. In *Proceedings of the International Conference on Machine Learning*, 2009.

[41] Y. Yue, T. Finley, F. Radlinski, and T. Joachims. A support vector method for optimizing average precision. In *Special Interest Group on Information Retrieval*, 2007.

